# Sparse Features for PCA-Like Linear Regression

**Christos Boutsidis**
Mathematical Sciences Department
IBM T. J. Watson Research Center
Yorktown Heights, New York
cboutsi@us.ibm.com

**Petros Drineas**
Computer Science Department
Rensselaer Polytechnic Institute
Troy, NY 12180
drinep@cs.rpi.edu

**Malik Magdon-Ismail**
Computer Science Department
Rensselaer Polytechnic Institute
Troy, NY 12180
magdon@cs.rpi.edu

## Abstract

Principal Components Analysis (PCA) is often used as a feature extraction proce-
dure. Given a matrix $X \in \mathbb{R}^{n \times d}$, whose rows represent $n$ data points with respect
to $d$ features, the top $k$ right singular vectors of $X$ (the so-called *eigenfeatures*),
are arbitrary linear combinations of all available features. The eigenfeatures are
very useful in data analysis, including the regularization of linear regression. En-
forcing sparsity on the eigenfeatures, i.e., forcing them to be linear combinations
of only a *small* number of actual features (as opposed to all available features), can
promote better generalization error and improve the interpretability of the eigen-
features. We present deterministic and randomized algorithms that construct such
sparse eigenfeatures while *provably* achieving in-sample performance comparable
to regularized linear regression. Our algorithms are relatively simple and practi-
cally efficient, and we demonstrate their performance on several data sets.

## 1 Introduction

Least-squares analysis was introduced by Gauss in 1795 and has since has bloomed into a staple of
the data analyst. Assume the usual setting with $n$ tuples $(\mathbf{x}_1, y_1), \ldots, (\mathbf{x}_n, y_n)$ in $\mathbb{R}^d$, where $\mathbf{x}_i$ are
points and $y_i$ are targets. The vector of regression weights $\mathbf{w}^* \in \mathbb{R}^d$ minimizes (over all $\mathbf{w} \in \mathbb{R}^d$)
the RMS in-sample error

$$\mathcal{E}(\mathbf{w}) = \sqrt{\sum_{i=1}^{n} (\mathbf{x}_i \cdot \mathbf{w} - y_i)^2} = \|X\mathbf{w} - \mathbf{y}\|_2.$$

In the above, $X \in \mathbb{R}^{n \times d}$ is the *data matrix* whose rows are the vectors $\mathbf{x}_i$ (i.e., $X_{ij} = \mathbf{x}_i[j]$); and,
$\mathbf{y} \in \mathbb{R}^n$ is the target vector (i.e., $\mathbf{y}[i] = y_i$). We will use the more convenient matrix formulation[1],
namely given $X$ and $\mathbf{y}$, we seek a vector $\mathbf{w}^*$ that minimizes $\|X\mathbf{w} - \mathbf{y}\|_2$. The minimal-norm vector
$\mathbf{w}^*$ can be computed via the Moore-Penrose pseudo-inverse of $X$: $\mathbf{w}^* = X^+\mathbf{y}$. Then, the optimal
in-sample error is equal to:

$$\mathcal{E}(\mathbf{w}^*) = \|\mathbf{y} - XX^+\mathbf{y}\|_2.$$

necessary.

When the data is noisy and X is ill-conditioned, $X^+$ becomes unstable to small perturbations and overfitting can become a serious problem. Practitioners deal with such situations by regularizing the regression. Popular regularization methods include, for example, the Lasso [28], Tikhonov regularization [17], and top-$k$ PCA regression or truncated SVD regularization [21]. In general, such methods are encouraging some form of parsimony, thereby reducing the number of effective degrees of freedom available to fit the data. Our focus is on top-$k$ PCA regression which can be viewed as regression onto the top-$k$ principal components, or, equivalently, the top-$k$ *eigenfeatures*. The eigenfeatures are the top-$k$ right singular vectors of X and are arbitrary linear combinations of all available input features. The question we tackle is whether one can efficiently extract *sparse eigenfeatures* (i.e., eigenfeatures that are linear combinations of only a small number of the available features) that have nearly the same performance as the top-$k$ eigenfeatures.

**Basic notation.** $A, B, \dots$ are matrices; $\mathbf{a}, \mathbf{b}, \dots$ are vectors; $i, j, \dots$ are integers; $I_n$ is the $n \times n$ identity matrix; $\mathbf{0}_{m \times n}$ is the $m \times n$ matrix of zeros; $\boldsymbol{e}_i$ is the standard basis (whose dimensionality will be clear from the context). For vectors, we use the Euclidean norm $\|\cdot\|_2$; for matrices, the Frobenius and the spectral norms: $\|X\|_F^2 = \sum_{i,j} X_{ij}^2$ and $\|X\|_2 = \sigma_1(X)$, i.e., the largest singular value of X.

**Top-$k$ PCA Regression.** Let $X = U\Sigma V^\mathsf{T}$ be the singular value decomposition of X, where U (resp. V) is the matrix of left (resp. right) singular vectors of X with singular values in the diagonal matrix $\Sigma$. For $k \leq d$, let $U_k, \Sigma_k$, and $V_k$ contain only the top-$k$ singular vectors and associated singular values. The best rank-$k$ reconstruction of X in the Frobenius norm can be obtained from this truncated singular value decomposition as $X_k = U_k \Sigma_k V_k^\mathsf{T}$. The $k$ right singular vectors in $V_k$ are called the top-$k$ *eigenfeatures*. The projections of the data points onto the top $k$ eigenfeatures are obtained by projecting the $\mathbf{x}_i$'s onto the columns of $V_k$ to obtain $F_k = XV_k = U\Sigma V^\mathsf{T} V_k = U_k \Sigma_k$. Now, each data point (row) in $F_k$ only has $k$ dimensions. Each column of $F_k$ contains a particular eigenfeature's value for every data point and is a linear combination of the columns of X.

The top-$k$ PCA regression uses $F_k$ as the data matrix and $\mathbf{y}$ as the target vector to produce regression weights $\mathbf{w}_k^* = F_k^+ \mathbf{y}$. The in-sample error of this $k$-dimensional regression is equal to

$$\|\mathbf{y} - F_k \mathbf{w}_k^*\|_2 = \|\mathbf{y} - F_k F_k^+ \mathbf{y}\|_2 = \|\mathbf{y} - U_k \Sigma_k \Sigma_k^{-1} U_k^\mathsf{T} \mathbf{y}\|_2 = \|\mathbf{y} - U_k U_k^\mathsf{T} \mathbf{y}\|_2.$$

The weights $\mathbf{w}_k^*$ are $k$-dimensional and cannot be applied to X, but the equivalent weights $V_k \mathbf{w}_k^*$ can be applied to X and they have the same in-sample error with respect to X:

$$\mathcal{E}(V_k \mathbf{w}_k^*) = \|\mathbf{y} - XV_k \mathbf{w}_k^*\|_2 = \|\mathbf{y} - F_k \mathbf{w}_k^*\|_2 = \|\mathbf{y} - U_k U_k^\mathsf{T} \mathbf{y}\|_2.$$

Hence, we will refer to both $\mathbf{w}_k^*$ and $V_k \mathbf{w}_k^*$ as the top-$k$ PCA regression weights (the dimension will make it clear which one we are talking about) and, for simplicity, we will overload $\mathbf{w}_k^*$ to refer to both these weight vectors (the dimension will make it clear which). In practice, $k$ is chosen to measure the "effective dimension" of the data, and, typically, $k \ll \text{rank}(X) = d$. One way to choose $k$ is so that $\|X - X_k\|_F \ll \sigma_k(X)$ (the "energy" in the $k$-th principal component is large compared to the energy in all smaller principal components). We do not argue the merits of top-$k$ PCA regression; we just note that top-$k$ PCA regression is a common tool for regularizing regression.

**Problem Formulation.** Given $X \in \mathbb{R}^{n \times d}$, $k$ (the number of target eigenfeatures for top-$k$ PCA regression), and $r > k$ (the sparsity parameter), we seek to extract a set of at most $k$ sparse eigenfeatures $\hat{V}_k$ which use at most $r$ of the actual dimensions. Let $\hat{F}_k = X\hat{V}_k \in \mathbb{R}^{n \times k}$ denote the matrix whose columns are the $k$ extracted sparse eigenfeatures, which are a linear combination of a set of at most $r$ actual features. Our goal is to obtain sparse features for which the vector of sparse regression weights $\hat{\mathbf{w}}_k = \hat{F}_k^+ \mathbf{y}$ results in an in-sample error $\|\mathbf{y} - \hat{F}_k \hat{F}_k^+ \mathbf{y}\|_2$ that is close to the top-$k$ PCA regression error $\|\mathbf{y} - F_k F_k^+ \mathbf{y}\|_2$. Just as with top-$k$ PCA regression, we can define the equivalent $d$-dimensional weights $\hat{V}_k \hat{\mathbf{w}}_k$; we will overload $\hat{\mathbf{w}}_k$ to refer to these weights as well.

Finally, we conclude by noting that while our discussion above has focused on simple linear regression, the problem can also be defined for multiple regression, where the vector $\mathbf{y}$ is replaced by a matrix $Y \in \mathbb{R}^{n \times \omega}$, with $\omega \geq 1$. The weight vector $\mathbf{w}$ becomes a weight matrix, W, where each column of W contains the weights from the regression of the corresponding column of Y onto the features. All our results hold in this general setting as well, and we will actually present our main contributions in the context of multiple regression.

## 2 Our contributions

Recall from our discussion at the end of the introduction that we will present all our results in the general setting, where the target vector $\mathbf{y}$ is replaced by a matrix $\mathbf{Y} \in \mathbb{R}^{n \times \omega}$. Our first theorem argues that there exists a polynomial-time deterministic algorithm that constructs a feature matrix $\hat{\mathbf{F}}_k \in \mathbb{R}^{n \times k}$, such that each feature (column of $\hat{\mathbf{F}}_k$) is a linear combination of *at most $r$* actual features (columns) from X and results in small in-sample error . Again, this should be contrasted with top-$k$ PCA regression, which constructs a feature matrix $\mathbf{F}_k$, such that each feature (column of $\mathbf{F}_k$) is a linear combination of *all* features (columns) in X. Our theorems argue that the in-sample error of our features is almost as good as the in-sample error of top-$k$ PCA regression, which uses dense features.

**Theorem 1** (Deterministic Feature Extraction). *Let* $\mathrm{X} \in \mathbb{R}^{n \times d}$ *and* $\mathrm{Y} \in \mathbb{R}^{n \times \omega}$ *be the input matrices in a multiple regression problem. Let* $k > 0$ *be a target rank for top-k PCA regression on* X *and* Y. *For any* $r > k$, *there exists an algorithm that constructs a feature matrix* $\hat{\mathrm{F}}_k = \mathrm{X}\hat{\mathrm{V}}_k \in \mathbb{R}^{n \times k}$, *such that every column of* $\hat{\mathrm{F}}_k$ *is a linear combination of (the same) at most $r$ columns of* X, *and*

$$\left\| \mathrm{Y} - \mathrm{X}\hat{\mathrm{W}}_k \right\|_F = \| \mathrm{Y} - \hat{\mathrm{F}}_k \hat{\mathrm{F}}_k^{+} \mathrm{Y} \|_F \leq \| \mathrm{Y} - \mathrm{X}\mathrm{W}_k^{*} \|_F + \left( 1 + \sqrt{\frac{9k}{r}} \right) \frac{\| \mathrm{X} - \mathrm{X}_k \|_F}{\sigma_k(\mathrm{X})} \| \mathrm{Y} \|_2.$$

*($\sigma_k(\mathrm{X})$ is the k-th singular value of* X.*) The running time of the proposed algorithm is* $T(V_k) + O\left(ndk + nrk^2\right)$, *where* $T(V_k)$ *is the time required to compute the matrix* $V_k$, *the top-k right singular vectors of* $X$.

Theorem 1 says that one can construct $k$ features with sparsity $O(k)$ and obtain a comparble regression error to that attained by the dense top-$k$ PCA features, up to additive term that is proportional to $\Delta_k = \| \mathrm{X} - \mathrm{X}_k \|_F / \sigma_k(\mathrm{X})$.

To construct the features satisfying the guarantees of the above theorem, we first employ the Algorithm DSF-Select (see Table 1 and Section 4.3) to select $r$ columns of X and form the matrix $\mathrm{C} \in \mathbb{R}^{n \times r}$. Now, let $\Pi_{\mathrm{C},k}(\mathrm{Y})$ denote the best rank-$k$ approximation (with respect to the Frobenius norm) to Y in the column-span of C. In other words, $\Pi_{\mathrm{C},k}(\mathrm{Y})$ is a rank-$k$ matrix that minimizes $\| \mathrm{Y} - \Pi_{\mathrm{C},k}(\mathrm{Y}) \|_F$ over all rank-$k$ matrices in the column-span of C. Efficient algorithms are known for computing $\Pi_{\mathrm{C},k}(\mathrm{X})$ and have been described in [2]. Given $\Pi_{\mathrm{C},k}(\mathrm{Y})$, the sparse eigenfeatures can be computed efficiently as follows: first, set $\Psi = \mathrm{C}^{+} \Pi_{\mathrm{C},k}(\mathrm{Y})$. Observe that

$$\mathrm{C}\Psi = \mathrm{C}\mathrm{C}^{+} \Pi_{\mathrm{C},k}(\mathrm{Y}) = \Pi_{\mathrm{C},k}(\mathrm{Y}).$$

The last equality follows because $\mathrm{C}\mathrm{C}^{+}$ projects onto the column span of C and $\Pi_{\mathrm{C},k}(\mathrm{Y})$ is already in the column span of C. $\Psi$ has rank at most $k$ because $\Pi_{\mathrm{C},k}(\mathrm{Y})$ has rank at most $k$. Let the SVD of $\Psi$ be $\Psi = \mathrm{U}_\psi \Sigma_\psi \mathrm{V}_\psi^{\mathrm{T}}$, and set $\hat{\mathrm{F}}_k = \mathrm{C}\mathrm{U}_\psi \Sigma_\psi \in \mathbb{R}^{n \times k}$. Clearly, each column of $\hat{\mathrm{F}}_k$ is a linear combination of (the same) at most $r$ columns of X (the columns in C). The sparse features themselves can also be obtained because $\hat{\mathrm{F}}_k = \mathrm{X}\hat{\mathrm{V}}_k$, so $\hat{\mathrm{V}}_k = \mathrm{X}^{+} \hat{\mathrm{F}}_k$.

To prove that $\hat{\mathrm{F}}_k$ are a good set of sparse features, we first relate the regression error from using $\hat{\mathrm{F}}_k$ to how well $\Pi_{\mathrm{C},k}(\mathrm{Y})$ approximates Y.

$$\| \mathrm{Y} - \Pi_{\mathrm{C},k}(\mathrm{Y}) \|_F = \| \mathrm{Y} - \mathrm{C}\Psi \|_F = \| \mathrm{Y} - \mathrm{C}\mathrm{U}_\psi \Sigma_\psi \mathrm{V}_\psi^{\mathrm{T}} \|_F = \| \mathrm{Y} - \hat{\mathrm{F}}_k \mathrm{V}_\psi^{\mathrm{T}} \|_F \geq \| \mathrm{Y} - \hat{\mathrm{F}}_k \hat{\mathrm{F}}_k^{+} \mathrm{Y} \|_F.$$

The last inequality follows because $\hat{\mathrm{F}}_k^{+} \mathrm{Y}$ are the optimal regression weights for the features $\hat{\mathrm{F}}_k$. The reverse inequality also holds because $\Pi_{\mathrm{C},k}(\mathrm{Y})$ is the best rank-$k$ approximation to Y in the column span of C. Thus,

$$\| \mathrm{Y} - \hat{\mathrm{F}}_k \hat{\mathrm{F}}_k^{+} \mathrm{Y} \|_F = \| \mathrm{Y} - \Pi_{\mathrm{C},k}(\mathrm{Y}) \|_F.$$

The upshot of the above discussion is that if we can find a matrix C consisting of columns of X for which $\| \mathrm{Y} - \Pi_{\mathrm{C},k}(\mathrm{Y}) \|_F$ is small, then we immediately have good sparse eigenfeatures. Indeed, all that remains to complete the proof of Theorem 1 is to bound $\| \mathrm{Y} - \Pi_{\mathrm{C},k}(\mathrm{Y}) \|_F$ for the columns C returned by the Algorithm DSF-Select.

Our second result employs the Algorithm RSF-Select (see Table 2 and Section 4.4) to select $r$ columns of X and again form the matrix $\mathrm{C} \in \mathbb{R}^{n \times r}$. One then proceeds to construct $\Pi_{\mathrm{C},k}(\mathrm{Y})$ and $\hat{\mathrm{F}}_k$ as described above. The advantage of this approach is simplicity, better efficiency and a slightly better error bound, at the expense of logarithmically worse sparsity.

**Theorem 2** (Randomized Feature Extraction). *Let $\mathrm{X} \in \mathbb{R}^{n \times d}$ and $\mathrm{Y} \in \mathbb{R}^{n \times \omega}$ be the input matrices in a multiple regression problem. Let $k > 0$ be a target rank for top-$k$ PCA regression on $\mathrm{X}$ and $\mathrm{Y}$. For any $r > 144k \ln(20k)$, there exists a randomized algorithm that constructs a feature matrix $\hat{\mathrm{F}}_k = \mathrm{X}\hat{\mathrm{V}}_k \in \mathbb{R}^{n \times k}$, such that every column of $\hat{\mathrm{F}}_k$ is a linear combination of at most $r$ columns of $\mathrm{X}$, and, with probability at least .7 (over random choices made in the algorithm),*

$$\left\| \mathrm{Y} - \mathrm{X}\hat{\mathrm{W}}_k \right\|_F = \| \mathrm{Y} - \hat{\mathrm{F}}_k\hat{\mathrm{F}}_k^+ \mathrm{Y} \|_F \leq \| \mathrm{Y} - \mathrm{X}\mathrm{W}_k^* \|_F + \sqrt{\frac{36k \ln(20k)}{r}} \frac{\|\mathrm{X} - \mathrm{X}_k\|_F}{\sigma_k(\mathrm{X})} \|\mathrm{Y}\|_2.$$

*The running time of the proposed algorithm is $T(\mathrm{V}_k) + O(dk + r \log r)$.*

## 3  Connections with prior work

A variant of our problem is the identification of a matrix C consisting of a small number (say $r$) columns of X such that the regression of Y onto C (as opposed to $k$ features from C) gives small in-sample error. This is the sparse approximation problem, where the number of non-zero weights in the regression vector is restricted to $r$. This problem is known to be NP-hard [25]. Sparse approximation has important applications and many approximation algorithms have been presented [29, 9, 30]; proposed algorithms are typically either greedy or are based on convex optimization relaxations of the objective. An important difference between sparse approximation and sparse PCA regression is that our goal is not to minimize the error under a sparsity constraint, but to match the top-$k$ PCA regularized regression under a sparsity constraint. We argue that it is possible to achieve a provably accurate sparse PCA-regression, i.e., use sparse features instead of dense ones.

If $\mathrm{X} = \mathrm{Y}$ (approximating X using the columns of X), then this is the column-based matrix reconstruction problem, which has received much attention in existing literature [16, 18, 14, 26, 5, 12, 20]. In this paper, we study the more general problem where $\mathrm{X} \neq \mathrm{Y}$, which turns out to be considerably more difficult.

Input sparseness is closely related to feature selection and automatic relevance determination. Research in this area is vast, and we refer the reader to [19] for a high-level view of the field. Again, the goal in this area is different than ours, namely they seek to reduce dimensionality and improve out-of-sample error. Our goal is to provide sparse PCA features that are almost as good as the exact principal components. While it is definitely the case that many methods outperform top-$k$ PCA regression, especially for $d \gg n$, this discussion is orthogonal to our work.

The closest result to ours in prior literature is the so-called rank-revealing QR (RRQR) factorization [8]. The authors use a QR-like decomposition to select exactly $k$ columns of X and compare their sparse solution vector $\hat{\mathbf{w}}_k$ with the top-$k$ PCA regularized solution $\mathbf{w}_k^*$. They show that

$$\|\mathbf{w}_k^* - \hat{\mathbf{w}}_k\|_2 \leq \sqrt{k(n-k)+1} \frac{\|\mathrm{X} - \mathrm{X}_k\|_2}{\sigma_k(\mathrm{X})} \Delta,$$

where $\Delta = 2\|\hat{\mathbf{w}}_k\|_2 + \|\mathbf{y} - \mathrm{X}\mathbf{w}_k^*\|_2 / \sigma_k(\mathrm{X})$. This bound is similar to our bound in Theorem 1, but only applies to $r = k$ and is considerably weaker. For example, $\sqrt{k(n-k)+1} \|\mathrm{X} - \mathrm{X}_k\|_2 \geq \sqrt{k} \|\mathrm{X} - \mathrm{X}_k\|_F$; note also that the dependence of the above bound on $1/\sigma_k(\mathrm{X})$ is generally worse than ours.

The importance of the right singular vectors in matrix reconstruction problems (including PCA) has been heavily studied in prior literature, going back to work by Jolliffe in 1972 [22]. The idea of sampling columns from a matrix X with probabilities that are derived from $\mathrm{V}_k^\mathsf{T}$ (as we do in Theorem 2) was introduced in [15] in order to construct *coresets* for regression problems by sampling data points (rows of the matrix X) as opposed to features (columns of the matrix X). Other prior work including [15, 13, 27, 6, 4] has employed variants of this sampling scheme; indeed, we borrow proof techniques from the above papers in our work. Finally, we note that our deterministic feature selection algorithm (Theorem 1) uses a sparsification tool developed in [2] for column based matrix reconstruction. This tool is a generalization of algorithms originally introduced in [1].

# 4 Our algorithms

Our algorithms emerge from the constructive proofs of Theorems 1 and 2. Both algorithms necessitate access to the right singular vectors of X, namely the matrix $V_k \in \mathbb{R}^{d \times k}$. In our experiments, we used PROPACK [23] in order to compute $V_k$ iteratively; PROPACK is a fast alternative to the exact SVD. Our first algorithm (DSF-Select) is deterministic, while the second algorithm (RSF-Select) is randomized, requiring logarithmically more columns to guarantee the theoretical bounds. Prior to describing our algorithms in detail, we will introduce useful notation on sampling and rescaling matrices as well as a matrix factorization lemma (Lemma 3) that will be critical in our proofs.

## 4.1 Sampling and rescaling matrices

Let $C \in \mathbb{R}^{n \times r}$ contain $r$ columns of $X \in \mathbb{R}^{n \times d}$. We can express the matrix C as $C = X\Omega$, where the *sampling* matrix $\Omega \in \mathbb{R}^{d \times r}$ is equal to $[\mathbf{e}_{i_1}, \ldots, \mathbf{e}_{i_r}]$ and $\mathbf{e}_i$ are standard basis vectors in $\mathbb{R}^d$. In our proofs, we will make use of $S \in \mathbb{R}^{r \times r}$, a diagonal *rescaling* matrix with positive entries on the diagonal. Our column selection algorithms return a sampling and a rescaling matrix, so that $X\Omega S$ contains a subset of rescaled columns from X. The rescaling is benign since it does not affect the span of the columns of $C = X\Omega$ and thus the quantity of interest, namely $\Pi_{C,k}(Y)$.

## 4.2 A structural result using matrix factorizations

We now present a matrix reconstruction lemma that will be the starting point for our algorithms. Let $Y \in \mathbb{R}^{n \times \omega}$ be a target matrix and let $X \in \mathbb{R}^{n \times d}$ be the basis matrix that we will use in order to reconstruct Y. More specifically, we seek a *sparse reconstruction* of Y from X, or, in other words, we would like to choose $r \ll d$ columns from X and form a matrix $C \in \mathbb{R}^{n \times r}$ such that $\|Y - \Pi_{C,k}(Y)\|_F$ is small. Let $Z \in \mathbb{R}^{d \times k}$ be an orthogonal matrix (i.e., $Z^T Z = I_k$), and express the matrix X as follows:

$$X = HZ^T + E,$$

where H is some matrix in $\mathbb{R}^{n \times k}$ and $E \in \mathbb{R}^{n \times d}$ is the residual error of the factorization. It is easy to prove that the Frobenius or spectral norm of E is minimized when $H = XZ$. Let $\Omega \in \mathbb{R}^{d \times r}$ and $S \in \mathbb{R}^{r \times r}$ be a sampling and a rescaling matrix respectively as defined in the previous section, and let $C = X\Omega \in \mathbb{R}^{n \times r}$. Then, the following lemma holds (see [3] for a detailed proof).

**Lemma 3** (Generalized Column Reconstruction)**.** *Using the above notation, if the rank of the matrix* $Z^T \Omega S$ *is equal to k, then*

$$\|Y - \Pi_{C,k}(Y)\|_F \leq \|Y - HH^+ Y\|_F + \|E\Omega S(Z^T \Omega S)^+ H^+ Y\|_F. \tag{1}$$

We now parse the above lemma carefully in order to understand its implications in our setting. For our goals, the matrix C essentially contains a subset of $r$ features from the data matrix X. Recall that $\Pi_{C,k}(Y)$ is the best rank-$k$ approximation to Y within the column space of C; and, the difference $Y - \Pi_{C,k}(Y)$ measures the error from performing regression using sparse eigenfeatures that are constructed as linear combinations of the columns of C. Moving to the right-hand side of eqn. (1), the two terms reflect a tradeoff between the accuracy of the reconstruction of Y using H and the error E in approximating X by the product $HZ^T$. Ideally, we would like to choose H so that Y can be accurately approximated and, at the same time, the matrix X is approximated by the product $HZ^T$ with small residual error E. In general, these two goals might be competing and a balance must be struck. Here, we focus on one extreme of this trade off, namely choosing Z so that the (Frobenius) norm of the matrix E is minimized. More specifically, since Z has rank $k$, the best choice for $HZ^T$ in order to minimize $\|E\|_F$ is $X_k$; then, $E = X - X_k$. Using the SVD of $X_k$, namely $X_k = U_k \Sigma_k V_k^T$, we apply Lemma 3 setting $H = U_k \Sigma_k$ and $Z = V_k$. The following corollary is immediate.

**Lemma 4** (Generalization of Lemma 7 in [2])**.** *Using the above notation, if the rank of the matrix* $V_k^T \Omega S$ *is equal to k, then*

$$\|Y - \Pi_{C,k}(Y)\|_F \leq \|Y - U_k U_k^T Y\|_F + \|(X - X_k)\Omega S(V_k^T \Omega S)^+ \Sigma_k^{-1} U_k^T Y\|_F.$$

Our main results will follow by carefully choosing $\Omega$ and S in order to control the right-hand side of the above inequality.

| Algorithm: DSF-Select | Algorithm: DetSampling |
|---|---|
| 1: **Input:** X, $k$, $r$.<br>2: **Output:** $r$ columns of X in C.<br>3: Compute $\mathbf{V}_k$ and<br><br>$$\mathbf{E} = \mathbf{X} - \mathbf{X}_k = \mathbf{X} - \mathbf{X}\mathbf{V}_k\mathbf{V}_k^{\mathsf{T}}.$$<br><br>4: Run DetSampling to construct sampling and rescaling matrices $\Omega$ and S:<br><br>$$[\Omega, \mathrm{S}] = \mathsf{DetSampling}(\mathbf{V}_k^{\mathsf{T}}, \mathbf{E}, r).$$<br><br>5: **Return** $\mathbf{C} = \mathbf{X}\Omega$. | 1: **Input:** $\mathbf{V}^{\mathsf{T}} = [\mathbf{v}_1, \ldots, \mathbf{v}_d]$, $\mathbf{A} = [\mathbf{a}_1, \ldots, \mathbf{a}_d]$, $r$.<br>2: **Output:** Sampling and rescaling matrices $[\Omega, \mathrm{S}]$.<br>3: Initialize $\mathbf{B}_0 = \mathbf{0}_{k\times k}$, $\Omega = \mathbf{0}_{d\times r}$, and $\mathrm{S} = \mathbf{0}_{r\times r}$.<br>4: **for** $\tau = 1$ **to** $r-1$ **do**<br>5:   Set $\mathrm{L}_\tau = \tau - \sqrt{rk}$.<br>6:   Pick index $i \in \{1, 2, ..., n\}$ and $t$ such that<br><br>$$U(\mathbf{a}_i) \le \frac{1}{t} \le L(\mathbf{v}_i, \mathbf{B}_{\tau-1}, \mathrm{L}_\tau).$$<br><br>7:   Update $\mathbf{B}_\tau = \mathbf{B}_{\tau-1} + t\mathbf{v}_i\mathbf{v}_i^{\mathsf{T}}$.<br>8:   Set $\Omega_{i\tau} = 1$ and $\mathrm{S}_{\tau\tau} = 1/\sqrt{t}$.<br>9: **end for**<br>10: **Return** $\Omega$ and S. |

Table 1: DSF-Select: Deterministic Sparse Feature Selection

### 4.3  DSF-Select**: Deterministic Sparse Feature Selection**

DSF-Select deterministically selects $r$ columns of the matrix X to form the matrix C (see Table 1 and note that the matrix $\mathbf{C} = \mathbf{X}\Omega$ might contain duplicate columns which can be removed without any loss in accuracy). The heart of DSF-Select is the subroutine DetSampling, a near-greedy algorithm which selects columns of $\mathbf{V}_k^{\mathsf{T}}$ iteratively to satisfy two criteria: the selected columns should form an approximately orthogonal basis for the columns of $\mathbf{V}_k^{\mathsf{T}}$ so that $(\mathbf{V}_k^{\mathsf{T}}\Omega\mathrm{S})^+$ is well-behaved; and $\mathbf{E}\Omega\mathrm{S}$ should also be well-behaved. These two properties will allow us to prove our results via Lemma 4. The implementation of the proposed algorithm is quite simple since it relies only on standard linear algebraic operations.

DetSampling takes as input two matrices: $\mathbf{V}^{\mathsf{T}} \in \mathbb{R}^{k\times d}$ (satisfying $\mathbf{V}^{\mathsf{T}}\mathbf{V} = \mathbf{I}_k$) and $\mathbf{A} \in \mathbb{R}^{n\times d}$. In order to describe the algorithm, it is convenient to view these two matrices as two sets of column vectors, $\mathbf{V}^{\mathsf{T}} = [\mathbf{v}_1, \ldots, \mathbf{v}_d]$ (satisfying $\sum_{i=1}^{d}\mathbf{v}_i\mathbf{v}_i^{\mathsf{T}} = \mathbf{I}_k$) and $\mathbf{A} = [\mathbf{a}_1, \ldots, \mathbf{a}_d]$. In DSF-Select we set $\mathbf{V}^{\mathsf{T}} = \mathbf{V}_k^{\mathsf{T}}$ and $\mathbf{A} = \mathbf{E} = \mathbf{X} - \mathbf{X}_k$. Given $k$ and $r$, the algorithm iterates from $\tau = 0$ up to $\tau = r-1$ and its main operation is to compute the functions $\phi(\mathrm{L}, \mathbf{B})$ and $L(\mathbf{v}, \mathbf{B}, \mathrm{L})$ that are defined as follows:

$$\phi(\mathrm{L}, \mathbf{B}) = \sum_{i=1}^{k}\frac{1}{\lambda_i - \mathrm{L}}, \qquad L(\mathbf{v}, \mathbf{B}, \mathrm{L}) = \frac{\mathbf{v}^{\mathsf{T}}(\mathbf{B} - (\mathrm{L}+1)\mathbf{I}_k)^{-2}\mathbf{v}}{\phi(\mathrm{L}+1, \mathbf{B}) - \phi(\mathrm{L}, \mathbf{B})} - \mathbf{v}^{\mathsf{T}}(\mathbf{B} - (\mathrm{L}+1)\mathbf{I}_k)^{-1}\mathbf{v}.$$

In the above, $\mathbf{B} \in \mathbb{R}^{k\times k}$ is a symmetric matrix with eigenvalues $\lambda_1, \ldots, \lambda_k$ and $\mathrm{L} \in \mathbb{R}$ is a parameter. We also define the function $U(\mathbf{a})$ for a vector $\mathbf{a} \in \mathbb{R}^n$ as follows:

$$U(\mathbf{a}) = \left(1 - \sqrt{\frac{k}{r}}\right)\frac{\mathbf{a}^{\mathsf{T}}\mathbf{a}}{\|\mathbf{A}\|_F^2}.$$

At every step $\tau$, the algorithm selects a column $\mathbf{a}_i$ such that $U(\mathbf{a}_i) \le L(\mathbf{v}_i, \mathbf{B}_{\tau-1}, \mathrm{L}_\tau)$; note that $\mathbf{B}_{\tau-1}$ is a $k \times k$ matrix which is also updated at every step of the algorithm (see Table 1). The existence of such a column is guaranteed by results in [1, 2].

It is worth noting that in practical implementations of the proposed algorithm, there might exist multiple columns which satisfy the above requirement. In our implementation we chose to break such ties arbitrarily. However, more careful and informed choices, such as breaking the ties in a way that makes maximum progress towards our objective, might result in considerable savings. This is indeed an interesting open problem.

The running time of our algorithm is dominated by the search for a column which satisfies $U(\mathbf{a}_i) \le L(\mathbf{v}_i, \mathbf{B}_{\tau-1}, \mathrm{L}_\tau)$. To compute the function $L$, we first need to compute $\phi(\mathrm{L}_\tau, \mathbf{B}_{\tau-1})$ (which necessitates the eigenvalues of $\mathbf{B}_{\tau-1}$) and then we need to compute the inverse of $\mathbf{B}_{\tau-1} - (\mathrm{L}+1)\mathbf{I}_k$. These computations need $O(k^3)$ time per iteration, for a total of $O(rk^3)$ time over all $r$ iterations. Now, in order to compute the function $L$ for each vector $\mathbf{v}_i$ for all $i = 1, \ldots, d$, we need an additional

| Algorithm: RSF-Select | Algorithm: RandSampling |
|---|---|
| 1: **Input:** X, $k$, $r$.<br>2: **Output:** $r$ columns of X in C.<br>3: Compute $V_k$.<br>4: Run RandSampling to construct sampling and rescaling matrices $\Omega$ and S:<br><br>$[\Omega, S] = \text{RandSampling}(V_k^{\mathsf{T}}, r).$<br><br>5: **Return** $C = X\Omega$. | 1: **Input:** $V^{\mathsf{T}} = [v_1, \ldots, v_d]$ and $r$.<br>2: **Output:** Sampling and rescaling matrices $[\Omega, S]$.<br>3: For $i = 1, ..., d$ compute probabilities<br><br>$p_i = \frac{1}{k}\|v_i\|_2^2.$<br><br>4: Initialize $\Omega = \mathbf{0}_{d\times r}$ and $S = \mathbf{0}_{r\times r}$.<br>5: **for** $\tau = 1$ **to** $r$ **do**<br>6: Select an index $i_\tau \in \{1, 2, ..., d\}$ where the probability of selecting index $i$ is equal to $p_i$.<br>7: Set $\Omega_{i_\tau \tau} = 1$ and $S_{\tau\tau} = 1/\sqrt{rp_{i_\tau}}$.<br>8: **end for**<br>9: **Return** $\Omega$ and S. |

Table 2: RSF-Select: Randomized Sparse Feature Selection

$O(dk^2)$ time per iteration; the total time for all $r$ iterations is $O(drk^2)$. Next, in order to compute the function $U$, we need to compute $a_i^{\mathsf{T}} a_i$ (for all $i = 1, \ldots, d$) which necessitates $O(\text{nnz}(A))$ time, where $\text{nnz}(A)$ is the number of non-zero elements of A. In our setting, $A = E \in \mathbb{R}^{n\times d}$, so the overall running time is $O(drk^2 + nd)$. In order to get the final running time we also need to account for the computation of $V_k$ and E.

The theoretical properties of DetSampling were analyzed in detail in [2], building on the original analysis of [1]. The following lemma from [2] summarizes important properties of $\Omega$.

**Lemma 5** ([2]). DetSampling *with inputs* $V^{\mathsf{T}}$ *and* A *returns a sampling matrix* $\Omega \in \mathbb{R}^{d\times r}$ *and a rescaling matrix* $S \in \mathbb{R}^{r\times r}$ *satisfying*

$$\|(V^{\mathsf{T}}\Omega S)^+\|_2 \le 1 - \sqrt{\frac{k}{r}}; \qquad \|A\Omega S\|_F \le \|A\|_F.$$

We apply Lemma 5 with $V = V_k^T$ and $A = E$ and we combine it with Lemma 4 to conclude the proof of Theorem 1; see [3] for details.

### 4.4 RSF-Select: **Randomized Sparse Feature Selection**

RSF-Select is a randomized algorithm that selects $r$ columns of the matrix X in order to form the matrix C (see Table 2). The main differences between RSF-Select and DSF-Select are two: first, RSF-Select only needs access to $V_k^{\mathsf{T}}$ and, second, RSF-Select uses a simple sampling procedure in order to select the columns of X to include in C. This sampling procedure is described in algorithm RandSampling and essentially selects columns of X with probabilities that depend on the norms of the columns of $V_k^{\mathsf{T}}$. Thus, RandSampling first computes a set of probabilities that are proportional to the norms of the columns of $V_k^{\mathsf{T}}$ and then samples $r$ columns of X in $r$ independent identical trials with replacement, where in each trial a column is sampled according to the computed probabilities. Note that a column could be selected multiple times. In terms of running time, and assuming that the matrix $V_k$ that contains the top $k$ right singular vectors of X has already been computed, the proposed algorithm needs $O(dk)$ time to compute the sampling probabilities and an additional $O(d + r\log r)$ time to sample $r$ columns from X. Similar to Lemma 5, we can prove analogous properties for the matrices $\Omega$ and S that are returned by algorithm RandSampling. Again, combining with Lemma 4 we can prove Theorem 2; see [3] for details.

## 5 Experiments

The goal of our experiments is to *illustrate* that our algorithms produce sparse features which perform as well in-sample as the top-$k$ PCA regression. It turns out that the out-of-sample performance is comparable (if not better in many cases, perhaps due to the sparsity) to top-$k$ PCA-regression.

| Data | $(n;d)$ | $k=5, r=k+1$ | | | | $k=5, r=2k$ | | | |
|---|---|---|---|---|---|---|---|---|---|
| | | $\mathbf{w}^*_k$ | $\hat{\mathbf{w}}^{\mathsf{DSF}}_k$ | $\hat{\mathbf{w}}^{\mathsf{RSF}}_k$ | $\hat{\mathbf{w}}^{\mathsf{rnd}}_k$ | $\mathbf{w}^*_k$ | $\hat{\mathbf{w}}^{\mathsf{DSF}}_k$ | $\hat{\mathbf{w}}^{\mathsf{RSF}}_k$ | $\hat{\mathbf{w}}^{\mathsf{rnd}}_k$ |
| Arcene | (100;10,000) | 0.93<br>0.99 | 0.88<br>**0.94** | 0.91<br>0.98 | 1.0<br>1.0 | 0.93<br>1.0 | 0.89<br>**0.97** | 0.86<br>0.98 | 1.0<br>1.0 |
| I-sphere | (351;34) | 0.57<br>0.58 | 0.52<br>**0.53** | 0.55<br>0.57 | 0.57<br>0.57 | 0.57<br>0.58 | 0.51<br>**0.54** | 0.52<br>0.55 | 0.56<br>0.56 |
| LibrasMov | (45;90) | 2.9<br>**3.3** | 2.9<br>3.6 | 3.1<br>3.7 | 3.7<br>3.7 | 2.9<br>**3.3** | 2.4<br>**3.3** | 2.6<br>3.6 | 3.6<br>3.6 |
| Madelon | (2,000;500) | 0.98<br>**0.98** | 0.98<br>**0.98** | 0.98<br>**0.98** | 1.0<br>1.0 | 0.98<br>**0.98** | 0.97<br>**0.98** | 0.97<br>**0.98** | 1.0<br>1.0 |
| HillVal | (606;100) | 0.68<br>0.68 | 0.66<br>**0.67** | 0.67<br>0.68 | 0.68<br>0.68 | 0.68<br>0.68 | 0.65<br>**0.67** | 0.67<br>0.69 | 0.69<br>0.69 |
| Spambase | (4601;57) | 0.30<br>**0.30** | 0.30<br>**0.30** | 0.31<br>**0.30** | 0.28<br>0.38 | 0.3<br>**0.3** | 0.3<br>**0.3** | 0.3<br>**0.3** | 0.25<br>0.35 |

Table 3: Comparison of DSF-select and RSF-select with top-$k$ PCA. The top entry in each cell is the in-sample error, and the bottom entry is the out-sample error. In bold is the method achieving the best out-sample error.

Compared to top-$k$ PCA, our algorithms are efficient and work well in practice, even better than the theoretical bounds suggest.

We present our findings in Table 3 using data sets from the UCI machine learning repository. We used a five-fold cross validation design with 1,000 random splits: we computed regression weights using 80% of the data and estimated out-sample error in the remaining 20% of the data. We set $k=5$ in the experiments (no attempt was made to optimize $k$). Table 3 shows the in- and out-sample error for four methods: top-$k$ PCA regression, $\mathbf{w}^*_k$; $r$-sparse features regression using DSF-select, $\hat{\mathbf{w}}^{\mathsf{DSF}}_k$; $r$-sparse features regression using RSF-select, $\hat{\mathbf{w}}^{\mathsf{RSF}}_k$; $r$-sparse features regression using $r$ random columns, $\hat{\mathbf{w}}^{\mathsf{rnd}}_k$.

# 6 Discussion

The top-$k$ PCA regression constructs "features" without looking at the targets – it is target-agnostic. So are all the algorithms we discussed here, as our goal was to compare with top-$k$ PCA. However, there is unexplored potential in Lemma 3. We only explored one extreme choice for the factorization, namely the minimization of some norm of the matrix E. Other choices, in particular non-target-agnostic choices, could prove considerably better. Such investigations are left for future work.

As mentioned when we discussed our deterministic algorithm, it will often be the case that in some steps of the greedy selection process, multiple columns could satisfy the criterion for selection. In such a situation, we are free to choose any one; we broke ties arbitrarily in our implementation, and even as is, the algorithm performed as well or better than top-$k$ PCA. However, we expect that breaking the ties so as to optimize the ultimate objective would yield considerable additional benefit; this would also be non-target-agnostic.

**Acknowledgments**

This work has been supported by two NSF CCF and DMS grants to Petros Drineas and Malik Magdon-Ismail.

## Footnotes

[1]For the sake of simplicity, we assume $d \leq n$ and rank $(X) = d$ in our exposition; neither assumption is

## References

[1] J. Batson, D. Spielman, and N. Srivastava. Twice-ramanujan sparsifiers. In *Proceedings of ACM STOC*, pages 255–262, 2009.

[2] C. Boutsidis, P. Drineas, and M. Magdon-Ismail. Near-optimal column based matrix reconstruction. In *Proceedings of IEEE FOCS*, 2011.

[3] C. Boutsidis, P. Drineas, and M. Magdon-Ismail. Sparse features for PCA-like linear regression. *manuscript*, 2011.

[4] C. Boutsidis and M. Magdon-Ismail. Deterministic feature selection for $k$-means clustering. *arXiv:1109.5664v1*, 2011.

[5] C. Boutsidis, M. W. Mahoney, and P. Drineas. An improved approximation algorithm for the column subset selection problem. In *Proceedings of ACM -SIAM SODA*, pages 968–977, 2009.

[6] C. Boutsidis, M. W. Mahoney, and P. Drineas. Unsupervised feature selection for the $k$-means clustering problem. In *Proceedings of NIPS*, 2009.

[7] J. Cadima and I. Jolliffe. Loadings and correlations in the interpretation of principal components. *Applied Statistics*, 22:203–214, 1995.

[8] T. Chan and P. Hansen. Some applications of the rank revealing QR factorization. *SIAM Journal on Scientific and Statistical Computing*, 13:727–741, 1992.

[9] A. Das and D. Kempe. Algorithms for subset selection in linear regression. In *Proceedings of ACM STOC*, 2008.

[10] A. Dasgupta, P. Drineas, B. Harb, R. Kumar, and M. W. Mahoney. Sampling algorithms and coresets for $L_p$ regression. In *Proceedings of ACM-SIAM SODA*, 2008.

[11] A. d'Aspremont, L. El Ghaoui, M. I. Jordan, and G. R. G. Lanckriet. A direct formulation for sparse PCA using semidefinite programming. In *Proceedings of NIPS*, 2004.

[12] A. Deshpande and L. Rademacher. Efficient volume sampling for row/column subset selection. In *Proceedings of ACM STOC*, 2010.

[13] P. Drineas, R. Kannan, and M. Mahoney. Fast Monte Carlo algorithms for matrices I: Approximating matrix multiplication. *SIAM Journal of Computing*, 36(1):132–157, 2006.

[14] P. Drineas, M. Mahoney, and S. Muthukrishnan. Polynomial time algorithm for column-row based relative-error low-rank matrix approximation. Technical Report 2006-04, DIMACS, March 2006.

[15] P. Drineas, M. Mahoney, and S. Muthukrishnan. Sampling algorithms for $\ell_2$ regression and applications. In *Proceedings of ACM-SIAM SODA*, pages 1127–1136, 2006.

[16] G. Golub. Numerical methods for solving linear least squares problems. *Numerische Mathematik*, 7:206–216, 1965.

[17] G. Golub, P. Hansen, and D. O'Leary. Tikhonov regularization and total least squares. *SIAM Journal on Matrix Analysis and Applications*, 21(1):185–194, 2000.

[18] M. Gu and S. Eisenstat. Efficient algorithms for computing a strong rank-revealing QR factorization. *SIAM Journal on Scientific Computing*, 17:848–869, 1996.

[19] I. Guyon and A. Elisseeff. Special issue on variable and feature selection. *Journal of Machine Learning Research*, 3, 2003.

[20] N. Halko, P. Martinsson, and J. Tropp. Finding structure with randomness: probabilistic algorithms for constructing approximate matrix decompositions. *SIAM Review*, 2011.

[21] P. Hansen. The truncated SVD as a method for regularization. *BIT Numerical Mathematics*, 27(4):534–553, 1987.

[22] I. Jolliffe. Discarding variables in Principal Component Analysis: asrtificial data. *Applied Statistics*, 21(2):160–173, 1972.

[23] R. Larsen. PROPACK: A software package for the symmetric eigenvalue problem and singular value problems on Lanczos and Lanczos bidiagonalization with partial reorthogonalization. http://soi.stanford.edu/∼rmunk/∼PROPACK/.

[24] B. Moghaddam, Y. Weiss, and S. Avidan. Spectral bounds for sparse PCA: exact and greedy algorithms. In *Proceedings of NIPS*, 2005.

[25] B. Natarajan. Sparse approximate solutions to linear systems. *SIAM Journal on Computing*, 24(2):227–234, 1995.

[26] M. Rudelson and R. Vershynin. Sampling from large matrices: An approach through geometric functional analysis. *Journal of the ACM*, 54, 2007.

[27] N. Srivastava and D. Spielman. Graph sparsifications by effective resistances. In *Proceedings of ACM STOC*, pages 563–568, 2008.

[28] R. Tibshirani. Regression shrinkage and selection via the lasso. *Journal of the Royal Statistical Society*, pages 267–288, 1996.

[29] J. Tropp. Greed is good: Algorithmic results for sparse approximation. *IEEE Transactions on Information Theory*, 50(10):2231–2242, 2004.

[30] T. Zhang. Generating a $d$-dimensional linear subspace efficiently. In *Adaptive forward-backward greedy algorithm for sparse learning with linear models*, 2008.

